# Canonical Time Warping
# for Alignment of Human Behavior

**Feng Zhou**
Robotics Institute
Carnegie Mellon University
www.f-zhou.com

**Fernando de la Torre**
Robotics Institute
Carnegie Mellon University
ftorre@cs.cmu.edu

## Abstract

Alignment of time series is an important problem to solve in many scientific disciplines. In particular, temporal alignment of two or more subjects performing similar activities is a challenging problem due to the large temporal scale difference between human actions as well as the inter/intra subject variability. In this paper we present canonical time warping (CTW), an extension of canonical correlation analysis (CCA) for spatio-temporal alignment of human motion between two subjects. CTW extends previous work on CCA in two ways: (i) it combines CCA with dynamic time warping (DTW), and (ii) it extends CCA by allowing local spatial deformations. We show CTW's effectiveness in three experiments: alignment of synthetic data, alignment of motion capture data of two subjects performing similar actions, and alignment of similar facial expressions made by two people. Our results demonstrate that CTW provides both visually and qualitatively better alignment than state-of-the-art techniques based on DTW.

## 1 Introduction

Temporal alignment of time series has been an active research topic in many scientific disciplines such as bioinformatics, text analysis, computer graphics, and computer vision. In particular, temporal alignment of human behavior is a fundamental step in many applications such as recognition [1], temporal segmentation [2] and synthesis of human motion [3]. For instance consider Fig. 1a which shows one subject walking with varying speed and different styles and Fig. 1b which shows two subjects reading the same text.

Previous work on alignment of human motion has been addressed mostly in the context of recognizing human activities and synthesizing realistic motion. Typically, some models such as hidden Markov models [4, 5, 6], weighted principal component analysis [7], independent component analysis [8, 9] or multi-linear models [10] are learned from training data and in the testing phase the time series is aligned w.r.t. the learned dynamic model. In the context of computer vision a key aspect for successful recognition of activities is building view-invariant representations. Junejo *et al*. [1] proposed a view-invariant descriptor for actions making use of the affinity matrix between time instances. Caspi and Irani [11] temporally aligned videos from two closely attached cameras. Rao et al. [12, 13] aligned trajectories of two moving points using constraints from the fundamental matrix. In the literature of computer graphics, Hsu *et al*. [3] proposed the iterative motion warping, a method that finds a spatio-temporal warping between two instances of motion captured data. In the context of data mining there have been several extensions of DTW [14] to align time series. Keogh and Pazzani [15] used derivatives of the original signal to improve alignment with DTW. Listgarten *et al*. [16] proposed continuous profile models, a probabilistic method for simultaneously aligning and normalizing sets of time series.

A relatively unexplored problem in behavioral analysis is the alignment between the motion of the body of face in two or more subjects (*e.g*., Fig. 1). Major challenges to solve human motion align-

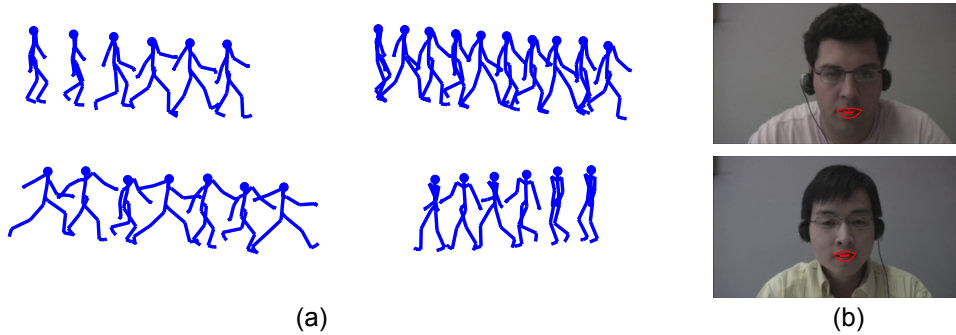

(a)            (b)

Figure 1: Temporal alignment of human behavior. (a) One person walking in normal pose, slow speed, another viewpoint and exaggerated steps (clockwise). (b) Two people reading the same text.

ment problems are: (i) allowing alignment between different sets of multidimensional features (*e.g.*, audio/video), (ii) introducing a feature selection or feature weighting mechanism to compensate for subject variability or irrelevant features and (iii) execution rate [17]. To solve these problems, this paper proposes canonical time warping (CTW) for accurate spatio-temporal alignment between two behavioral time series. We pose the problem as finding the temporal alignment that maximizes the spatial correlation between two behavioral samples coming from two subjects. To accommodate for subject variability and take into account the difference in the dimensionally of the signals, CTW uses CCA as a measure of spatial alignment. To allow temporal changes CTW incorporates DTW. CTW extends DTW by adding a feature weighting mechanism that is able to align signals of different dimensionality. CTW also extends CCA by incorporating time warping and allowing local spatial transformations.

The remainder of the paper is organized as follows. Section 2 reviews related work on dynamic time warping and canonical correlation analysis. Section 3 describes the new CTW algorithm. Section 4 extends CTW to take into account local transformations. Section 5 provides experimental results.

## 2   Previous work

This section describes previous work on canonical correlation analysis and dynamic time warping.

### 2.1   Canonical correlation analysis

Canonical correlation analysis (CCA) [18] is a technique to extract common features from a pair of multivariate data. CCA identifies relationships between two sets of variables by finding the linear combinations of the variables in the first set[1] ($\mathbf{X} \in \mathbb{R}^{d_x \times n}$) that are most correlated with the linear combinations of the variables in the second set ($\mathbf{Y} \in \mathbb{R}^{d_y \times n}$). Assuming zero-mean data, CCA finds a combination of the original variables that minimizes:

$$J_{cca}(\mathbf{V}_x, \mathbf{V}_y) = \|\mathbf{V}_x^T \mathbf{X} - \mathbf{V}_y^T \mathbf{Y}\|_F^2 \quad \text{s.t.} \quad \mathbf{V}_x^T \mathbf{X} \mathbf{X}^T \mathbf{V}_x = \mathbf{V}_y^T \mathbf{Y} \mathbf{Y}^T \mathbf{V}_y = \mathbf{I}_b, \tag{1}$$

where $\mathbf{V}_x \in \mathbb{R}^{d_x \times b}$ is the projection matrix for $\mathbf{X}$ (similarly for $\mathbf{V}_y$). The pair of canonical variates ($\mathbf{v}_x^T \mathbf{X}$, $\mathbf{v}_y^T \mathbf{Y}$) is uncorrelated with other canonical variates of lower order. Each successive canonical variate pair achieves the maximum correlation orthogonal to the preceding pairs. Eq. 1 has a closed form solution in terms of a generalized eigenvalue problem. See [19] for a unification of several component analysis methods and a review of numerical techniques to efficiently solve the generalized eigenvalue problems.

In computer vision, CCA has been used for matching sets of images in problems such as activity recognition from video [20] and activity correlation from cameras [21]. Recently, Fisher *et al*. [22]

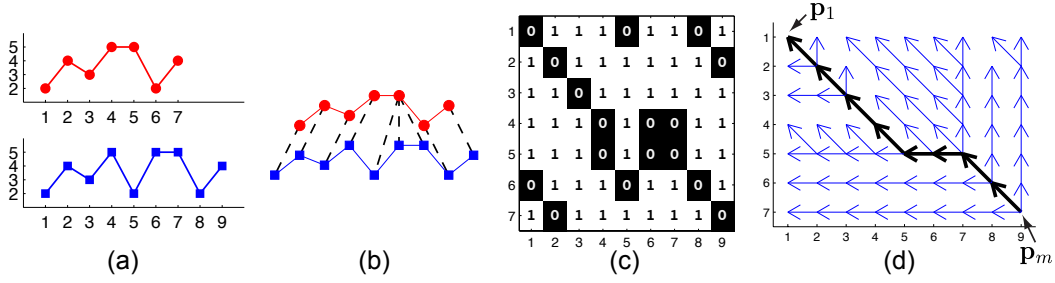

Figure 2: Dynamic time warping. (a) 1-D time series ($n_x = 7$ and $n_y = 9$). (b) DTW alignment. (c) Binary distance matrix. (d) Policy function at each node, where $\uparrow, \searrow, \leftarrow$ denote the policy, $\pi(\mathbf{p}_t) = [1,0]^T, [1,1]^T, [0,1]^T$, respectively. The optimal alignment path is denoted in bold.

proposed an extension of CCA with parameterized warping functions to align protein expressions. The learned warping function is a linear combination of hyperbolic tangent functions with non-negative coefficients, ensuring monotonicity. Unlike our method, the warping function is unable to deal with feature weighting.

## 2.2 Dynamic time warping

Given two time series, $\mathbf{X} = [\mathbf{x}_1, \mathbf{x}_2, \cdots, \mathbf{x}_{n_x}] \in \mathbb{R}^{d \times n_x}$ and $\mathbf{Y} = [\mathbf{y}_1, \mathbf{y}_2, \cdots, \mathbf{y}_{n_y}] \in \mathbb{R}^{d \times n_y}$, dynamic time warping [14] is a technique to optimally align the samples of $\mathbf{X}$ and $\mathbf{Y}$ such that the following sum-of-squares cost is minimized:

$$J_{dtw}(\mathbf{P}) = \sum_{t=1}^{m} \|\mathbf{x}_{p_t^x} - \mathbf{y}_{p_t^y}\|^2, \tag{2}$$

where $m$ is the number of indexes (or steps) needed to align both signals. The correspondence matrix $\mathbf{P}$ can be parameterized by a pair of path vectors, $\mathbf{P} = [\mathbf{p}^x, \mathbf{p}^y]^T \in \mathbb{R}^{2 \times m}$, in which $\mathbf{p}^x \in \{1:n_x\}^{m \times 1}$ and $\mathbf{p}^y \in \{1:n_y\}^{m \times 1}$ denote the composition of alignment in frames. For instance, the $i^{th}$ frame in $\mathbf{X}$ and the $j^{th}$ frame in $\mathbf{Y}$ are aligned iff there exists $\mathbf{p}_t = [p_t^x, p_t^y]^T = [i, j]^T$ for some $t$. $\mathbf{P}$ has to satisfy three additional constraints: boundary condition ($\mathbf{p}_1 \equiv [1,1]^T$ and $\mathbf{p}_m \equiv [n_x, n_y]^T$), continuity ($\mathbf{0} \leq \mathbf{p}_t - \mathbf{p}_{t-1} \leq \mathbf{1}$) and monotonicity ($t_1 \geq t_2 \Rightarrow \mathbf{p}_{t_1} - \mathbf{p}_{t_2} \geq \mathbf{0}$).

Although the number of possible ways to align $\mathbf{X}$ and $\mathbf{Y}$ is exponential in $n_x$ and $n_y$, dynamic programming [23] offers an efficient ($O(n_x n_y)$) approach to minimize $J_{dtw}$ using Bellman's equation:

$$L^*(\mathbf{p}_t) = \min_{\pi(\mathbf{p}_t)} \|\mathbf{x}_{p_t^x} - \mathbf{y}_{p_t^y}\|^2 + L^*(\mathbf{p}_{t+1}), \tag{3}$$

where the cost-to-go value function, $L^*(\mathbf{p}_t)$, represents the remaining cost starting at $t^{th}$ step to be incurred following the optimum policy $\pi^*$. The policy function, $\pi : \{1:n_x\} \times \{1:n_y\} \to \{[1,0]^T, [0,1]^T, [1,1]^T\}$, defines the deterministic transition between consecutive steps, $\mathbf{p}_{t+1} = \mathbf{p}_t + \pi(\mathbf{p}_t)$. Once the policy queue is known, the alignment steps can be recursively constructed from the starting point, $\mathbf{p}_1 = [1,1]^T$. Fig. 2 shows an example of DTW to align two 1-D time series.

## 3 Canonical time warping (CTW)

This section describes the energy function and optimization strategies for CTW.

### 3.1 Energy function for CTW

In order to have a compact and compressible energy function for CTW, it is important to notice that Eq. 2 can be rewritten as:

$$J_{dtw}(\mathbf{W}_x, \mathbf{W}_y) = \sum_{i=1}^{n_x} \sum_{j=1}^{n_y} \mathbf{w}_i^{x T} \mathbf{w}_j^y \|\mathbf{x}_i - \mathbf{y}_j\|^2 = \|\mathbf{X}\mathbf{W}_x^T - \mathbf{Y}\mathbf{W}_y^T\|_F^2, \tag{4}$$

where $\mathbf{W}_x \in \{0,1\}^{m \times n_x}$, $\mathbf{W}_y \in \{0,1\}^{m \times n_y}$ are binary selection matrices that need to be inferred to align $\mathbf{X}$ and $\mathbf{Y}$. In Eq. 4 the matrices $\mathbf{W}_x$ and $\mathbf{W}_y$ encode the alignment path. For instance,

$w_{tp_t^x}^x = w_{tp_t^y}^y = 1$ assigns correspondence between the $p_t^{x\,th}$ frame in $\mathbf{X}$ and $p_t^{y\,th}$ frame in $\mathbf{Y}$. For convenience, we denote, $\mathbf{D}_x = \mathbf{W}_x^T \mathbf{W}_x$, $\mathbf{D}_y = \mathbf{W}_y^T \mathbf{W}_y$ and $\mathbf{W} = \mathbf{W}_x^T \mathbf{W}_y$. Observe that Eq. 4 is very similar to the CCA's objective (Eq. 1). CCA applies a linear transformation to the rows (features), while DTW applies binary transformations to the columns (time).

In order to accommodate for differences in style and subject variability, add a feature selection mechanism, and reduce the dimensionality of the signals, CTW adds a linear transformation $(\mathbf{V}_x^T, \mathbf{V}_y^T)$ (as CCA) to the least-squares form of DTW (Eq. 4). Moreover, this transformation allows aligning temporal signals with different dimensionality (*e.g.*, video and motion capture). CTW combines DTW and CCA by minimizing:

$$J_{ctw}(\mathbf{W}_x, \mathbf{W}_y, \mathbf{V}_x, \mathbf{V}_y) = \|\mathbf{V}_x^T \mathbf{X} \mathbf{W}_x^T - \mathbf{V}_y^T \mathbf{Y} \mathbf{W}_y^T\|_F^2, \tag{5}$$

where $\mathbf{V}_x \in \mathbb{R}^{d_x \times b}$, $\mathbf{V}_y \in \mathbb{R}^{d_y \times b}$, $b \leq \min(d_x, d_y)$ parameterize the spatial warping by projecting the sequences into the same coordinate system. $\mathbf{W}_x$ and $\mathbf{W}_y$ warp the signal in time to achieve optimum temporal alignment. Similar to CCA, to make CTW invariant to translation, rotation and scaling, we impose the following constraints: (i) $\mathbf{X}\mathbf{W}_x^T \mathbf{1}_m = \mathbf{0}_{d_x}$, $\mathbf{Y}\mathbf{W}_y^T \mathbf{1}_m = \mathbf{0}_{d_y}$, (ii) $\mathbf{V}_x^T \mathbf{X} \mathbf{D}_x \mathbf{X}^T \mathbf{V}_x = \mathbf{V}_y^T \mathbf{Y} \mathbf{D}_y \mathbf{Y}^T \mathbf{V}_y = \mathbf{I}_b$ and (iii) $\mathbf{V}_x^T \mathbf{X} \mathbf{W} \mathbf{Y}^T \mathbf{V}_y$ to be a diagonal matrix. Eq. 5 is the main contribution of this paper. CTW is a direct and clean extension of CCA and DTW to align two signals $\mathbf{X}$ and $\mathbf{Y}$ in space and time. It extends previous work on CCA by adding temporal alignment and on DTW by allowing a feature selection and dimensionality reduction mechanism for aligning signals of different dimensions.

### 3.2 Optimization for CTW

---
**Algorithm 1**: *Canonical Time Warping*

---
**input** : $\mathbf{X}, \mathbf{Y}$
**output**: $\mathbf{V}_x, \mathbf{V}_y, \mathbf{W}_x, \mathbf{W}_y$

**begin**
    *Initialize* $\mathbf{V}_x = \mathbf{I}_{d_x}, \mathbf{V}_y = \mathbf{I}_{d_y}$
    **repeat**
        *Use dynamic programming to compute,* $\mathbf{W}_x, \mathbf{W}_y$, *for aligning the sequences,* $\mathbf{V}_x^T \mathbf{X}, \mathbf{V}_y^T \mathbf{Y}$
        *Set columns of,* $\mathbf{V}^T = [\mathbf{V}_x^T, \mathbf{V}_y^T]$, *be the leading $b$ generalized eigenvectors of:*

$$\begin{bmatrix} \mathbf{0} & \mathbf{X}\mathbf{W}\mathbf{Y}^T \\ \mathbf{Y}\mathbf{W}^T\mathbf{X}^T & \mathbf{0} \end{bmatrix} \mathbf{V} = \begin{bmatrix} \mathbf{X}\mathbf{D}_x\mathbf{X}^T & \mathbf{0} \\ \mathbf{0} & \mathbf{Y}\mathbf{D}_y\mathbf{Y}^T \end{bmatrix} \mathbf{V}\Lambda$$

    **until** $J_{ctw}$ *converges*
**end**

---

Optimizing $J_{ctw}$ is a non-convex optimization problem with respect to the alignment matrices $(\mathbf{W}_x, \mathbf{W}_y)$ and projection matrices $(\mathbf{V}_x, \mathbf{V}_y)$. We alternate between solving for $\mathbf{W}_x, \mathbf{W}_y$ using DTW, and optimally computing the spatial projections using CCA. These steps monotonically decrease $J_{ctw}$ and since the function is bounded below it will converge to a critical point.

Alg. 1 illustrates the optimization process (*e.g.*, Fig. 3e). The algorithm starts by initializing $\mathbf{V}_x$ and $\mathbf{V}_y$ with identity matrices. Alternatively, PCA can be applied independently to each set, and used as initial estimation of $\mathbf{V}_x$ and $\mathbf{V}_y$ if $d_x \neq d_y$. In the case of high-dimensional data, the generalized eigenvalue problem is solved by regularizing the covariance matrices adding a scaled identity matrix. The dimension $b$ is selected to preserve $90\%$ of the total correlation. We consider the algorithm to converge when the difference between two consecutive values of $J_{ctw}$ is small.

## 4 Local canonical time warping (LCTW)

In the previous section we have illustrated how CTW can align in space and time two time series of different dimensionality. However, there are many situations (*e.g.*, aligning long sequences) where a global transformation of the whole time series is not accurate. For these cases, local models have been shown to provide better performance [3, 24, 25]. This section extends CTW by allowing multiple local spatial deformations.

## 4.1 Energy function for LCTW

Let us assume that the spatial transformation for each frame in $\mathbf{X}$ and $\mathbf{Y}$ can be model as a linear combination of $k_x$ or $k_y$ bases. Let be $\mathbf{V}_x = [\mathbf{V}_1^{x\,T}, \cdots, \mathbf{V}_{k_x}^{x\,T}]^T \in \mathbb{R}^{k_x d_x \times b}$, $\mathbf{V}_y = [\mathbf{V}_1^{y\,T}, \cdots, \mathbf{V}_{k_y}^{y\,T}]^T \in \mathbb{R}^{k_y d_y \times b}$ and $b \leq \min(k_x d_x, k_y d_y)$. CTW allows for a more flexible spatial warping by minimizing:

$$J_{lctw}(\mathbf{W}_x, \mathbf{W}_y, \mathbf{V}_x, \mathbf{V}_y, \mathbf{R}_x, \mathbf{R}_y) \tag{6}$$

$$= \sum_{i=1}^{n_x} \sum_{j=1}^{n_y} \mathbf{w}_i^{x\,T} \mathbf{w}_j^y \| \Big( \sum_{c_x=1}^{k_x} r_{ic_x}^x \mathbf{V}_{c_x}^{x\,T} \Big) \mathbf{x}_i - \Big( \sum_{c_y=1}^{k_y} r_{jc_y}^y \mathbf{V}_{c_y}^{y\,T} \Big) \mathbf{y}_j \|^2 + \sum_{c_x=1}^{k_x} \| \mathbf{F}_x \mathbf{r}_{c_x}^x \|^2 + \sum_{c_y=1}^{k_y} \| \mathbf{F}_y \mathbf{r}_{c_y}^y \|^2$$

$$= \| \mathbf{V}_x^T \Big[ (\mathbf{1}_{k_x} \otimes \mathbf{X}) \circ (\mathbf{R}_x^T \otimes \mathbf{1}_{d_x}) \Big] \mathbf{W}_x^T - \mathbf{V}_y^T \Big[ (\mathbf{1}_{k_y} \otimes \mathbf{Y}) \circ (\mathbf{R}_y^T \otimes \mathbf{1}_{d_y}) \Big] \mathbf{W}_y^T \|_F^2 + \| \mathbf{F}_x \mathbf{R}_x \|_F^2 + \| \mathbf{F}_y \mathbf{R}_y \|_F^2,$$

where $\mathbf{R}_x \in \mathbb{R}^{n_x \times k_x}, \mathbf{R}_y \in \mathbb{R}^{n_y \times k_y}$ are the weighting matrices. $r_{ic_x}^x$ denotes the coefficient (or weight) of the $c_x^{th}$ basis for the $i^{th}$ frame of $\mathbf{X}$ (similarly for $r_{jc_y}^y$). We further constrain the weights to be positive (*i.e.*, $\mathbf{R}_x, \mathbf{R}_y \geq \mathbf{0}$) and the sum of weights to be one (*i.e.*, $\mathbf{R}_x \mathbf{1}_{k_x} = \mathbf{1}_{n_x}, \mathbf{R}_y \mathbf{1}_{k_y} = \mathbf{1}_{n_y}$) for each frame. The last two regularization terms, $\mathbf{F}_x \in \mathbb{R}^{n_x \times n_x}, \mathbf{F}_y \in \mathbb{R}^{n_y \times n_y}$, are $1^{st}$ order differential operators of $\mathbf{r}_{c_x}^x \in \mathbb{R}^{n_x \times 1}, \mathbf{r}_{c_y}^y \in \mathbb{R}^{n_y \times 1}$, encouraging smooth solutions over time. Observe that $J_{ctw}$ is a special case of $J_{lctw}$ when $k_x = k_y = 1$.

## 4.2 Optimization for LCTW

---

**Algorithm 2**: *Local Canonical Time Warping*

---
**input** : $\mathbf{X}, \mathbf{Y}$
**output**: $\mathbf{W}_x, \mathbf{W}_y, \mathbf{V}_x, \mathbf{V}_y, \mathbf{R}_x, \mathbf{R}_y$
**begin**

   *Initialize,*

$$\mathbf{V}_x = \mathbf{1}_{k_x} \otimes \mathbf{I}_{d_x}, \quad \mathbf{V}_y = \mathbf{1}_{k_y} \otimes \mathbf{I}_{d_y}$$

$$r_{ic_x}^x = 1 \text{ for } \lfloor \frac{(c_x-1)n_x}{k_x} \rfloor < i \leq \lfloor \frac{c_x n_x}{k_x} \rfloor, \quad r_{jc_y}^y = 1 \text{ for } \lfloor \frac{(c_y-1)n_y}{k_y} \rfloor < j \leq \lfloor \frac{c_y n_y}{k_y} \rfloor$$

   **repeat**

      *Denote,*

$$\mathbf{Z}_x = (\mathbf{1}_{k_x} \otimes \mathbf{X}) \circ (\mathbf{R}_x^T \otimes \mathbf{1}_{d_x}), \quad \mathbf{Z}_y = (\mathbf{1}_{k_y} \otimes \mathbf{Y}) \circ (\mathbf{R}_y^T \otimes \mathbf{1}_{d_y})$$

$$\mathbf{Q}_x = \mathbf{V}_x^T(\mathbf{I}_{k_x} \otimes \mathbf{X}), \quad \mathbf{Q}_y = \mathbf{V}_y^T(\mathbf{I}_{k_y} \otimes \mathbf{Y})$$

      *Use dynamic programming to compute, $\mathbf{W}_x, \mathbf{W}_y$, between the sequences, $\mathbf{V}_x^T \mathbf{Z}_x, \mathbf{V}_y^T \mathbf{Z}_y$*
      *Set columns of, $\mathbf{V}^T = [\mathbf{V}_x^T, \mathbf{V}_y^T]$, be the leading $b$ generalized eigenvectors,*

$$\begin{bmatrix} \mathbf{0} & \mathbf{Z}_x \mathbf{W} \mathbf{Z}_y^T \\ \mathbf{Z}_y \mathbf{W}^T \mathbf{Z}_x^T & \mathbf{0} \end{bmatrix} \mathbf{V} = \begin{bmatrix} \mathbf{Z}_x \mathbf{D}_x \mathbf{Z}_x^T & \mathbf{0} \\ \mathbf{0} & \mathbf{Z}_y \mathbf{D}_y \mathbf{Z}_y^T \end{bmatrix} \mathbf{V}\Lambda$$

      *Set, $\mathbf{r} = \text{Vec}([\mathbf{R}_x, \mathbf{R}_y])$, be the solution of the quadratic programming problem,*

$$\min_{\mathbf{r}} \quad \mathbf{r}^T \begin{bmatrix} \mathbf{1}_{k_x \times k_x} \otimes \mathbf{D}_x \circ \mathbf{Q}_x^T \mathbf{Q}_x + \mathbf{I}_{k_x} \otimes \mathbf{F}_x^T \mathbf{F}_x & -\mathbf{1}_{k_x \times k_y} \otimes \mathbf{W} \circ \mathbf{Q}_x^T \mathbf{Q}_y \\ -\mathbf{1}_{k_y \times k_x} \otimes \mathbf{W}^T \circ \mathbf{Q}_y^T \mathbf{Q}_x & \mathbf{1}_{k_y \times k_y} \otimes \mathbf{D}_y \circ \mathbf{Q}_y^T \mathbf{Q}_y + \mathbf{I}_{k_y} \otimes \mathbf{F}_y^T \mathbf{F}_y \end{bmatrix} \mathbf{r}$$

$$\text{s.t.} \begin{bmatrix} \mathbf{1}_{k_x}^T \otimes \mathbf{I}_{n_x} & \mathbf{0} \\ \mathbf{0} & \mathbf{1}_{k_y}^T \otimes \mathbf{I}_{n_y} \end{bmatrix} \mathbf{r} = \mathbf{1}_{n_x + n_y} \quad \mathbf{r} \geq \mathbf{0}_{n_x k_x + n_y k_y}$$

   **until** $J_{lctw}$ *converges*
**end**

---

As in the case of CTW, we use an alternating scheme for optimizing $J_{lctw}$, which is summarized in Alg. 2. In the initialization, we assume that each time series is divided into $k_x$ or $k_y$ equal parts, being the identity matrix the starting value for $\mathbf{V}_{c_x}^x, \mathbf{V}_{c_y}^y$ and block structure matrices for $\mathbf{R}_x, \mathbf{R}_y$.

The main difference between the alternating scheme of Alg. 1 and Alg. 2 is that the alternation step is no longer unique. For instance, when fixing $\mathbf{V}_x, \mathbf{V}_y$, one can optimize either $\mathbf{W}_x, \mathbf{W}_y$ or $\mathbf{R}_x, \mathbf{R}_y$. Consider a simple example of warping $\sin(t_1)$ towards $\sin(t_2)$, one could shift the first sequence along time axis by $\delta_t = t_2 - t_1$ or do the linear transformation, $a_{t_1}\sin(t_1) + b_{t_1}$, where $a_{t_1} = \cos(t_2 - t_1)$ and $b_{t_1} = \cos(t_1)\sin(t_2 - t_1)$. In order to better control the trade-off between time warping and spatial transformation, we propose a stochastic selection process. Let us denote $p_{w|v}$ the conditional probability of optimizing $\mathbf{W}$ when fixing $\mathbf{V}$. Given the prior probabilities $[p_w, p_v, p_r]$, we can derive the conditional probabilities using Bayes' theorem and the fact that, $[p_{r|w}, p_{r|v}, p_{v|r}] = 1 - [p_{v|w}, p_{w|v}, p_{w|r}]$. $[p_{v|w}, p_{w|v}, p_{w|r}]^T = \mathbf{A}^{-1}\mathbf{b}$ , where $\mathbf{A} = \begin{bmatrix} p_w & -p_v & 0 \\ p_w & 0 & p_r \\ 0 & -p_v & p_r \end{bmatrix}$ and $\mathbf{b} = \begin{bmatrix} 0 \\ p_w \\ -p_v + p_r \end{bmatrix}$. Fig. 3f (right-lower corner) shows the optimization strategy, $p_w = .5, p_v = .3, p_r = .2$, where the time warping process is more often optimized.

## 5  Experiments

This section demonstrates the benefits of CTW and LCTW against state-of-the-art DTW approaches to align synthetic data, motion capture data of two subjects performing similar actions, and similar facial expressions made by two people.

### 5.1  Synthetic data

In the first experiment we synthetically generated two spatio-temporal signals (3-D in space and 1-D in time) to evaluate the performance of CTW and LCTW. The first two spatial dimensions and the time dimension are generated as follows: $\mathbf{X} = \mathbf{U}_x^T \mathbf{Z} \mathbf{M}_x^T$ and $\mathbf{Y} = \mathbf{U}_y^T \mathbf{Z} \mathbf{M}_y^T$, where $\mathbf{Z} \in \mathbb{R}^{2 \times m}$ is a curve in two dimensions (Fig. 3a). $\mathbf{U}_x, \mathbf{U}_y \in \mathbb{R}^{2 \times 2}$ are randomly generated affine transformation matrices for the spatial warping and $\mathbf{M}_x \in \mathbb{R}^{n_x \times m}, \mathbf{M}_y \in \mathbb{R}^{n_y \times m}, m \geq \max(n_x, n_y)$ are randomly generated matrices for time warping[2]. The third spatial dimension is generated by adding a $(1 \times n_x)$ or $(1 \times n_y)$ extra row to $\mathbf{X}$ and $\mathbf{Y}$ respectively, with zero-mean Gaussian noise (see Fig. 3a-b).

We compared the performance of CTW and LCTW against three other methods: (i) dynamic time warping (DTW) [14], (ii) derivative dynamic time warping (DDTW) [15] and (iii) iterative time warping (IMW) [3]. Recall that in the case of synthetic data we know the ground truth alignment matrix $\mathbf{W}_{truth} = \mathbf{M}_x \mathbf{M}_y^T$. The error between the ground truth and a given alignment $\mathbf{W}_{alg}$ is computed by the area enclosed between both paths (see Fig. 3g).

Fig. 3c-f show the spatial warping estimated by each algorithm. DDTW (Fig. 3c) cannot deal with this example because the feature derivatives do not capture well the structure of the sequence. IMW (Fig. 3d) warps one sequence towards the other by translating and re-scaling each frame in each dimension. Fig. 3h shows the testing error (space and time) for 100 new generated time series. As it can be observed CTW and LCTW obtain the best performance. IMW has more parameters ($O(dn)$) than CTW ($O(db)$) and LCTW ($O(kdb + kn)$), and hence IMW is more prone to overfitting. IMW tries to fit the noisy dimension ($3^{rd}$ spatial component) biasing alignment in time (Fig. 3g), whereas CTW and LCTW have a feature selection mechanism which effectively cancels the third dimension. Observe that the null space for the projection matrices in CTW is $\mathbf{v}_x^T = [.002, .001, -.067]^T$, $\mathbf{v}_y^T = [-.002, -.001, -.071]^T$.

### 5.2  Motion capture data

In the second experiment we apply CTW and LCTW to align human motion with similar behavior. The motion capture data is taken from the CMU-Multimodal Activity Database [26]. We selected a pair of sub-sequences from subject 1 and subject 3 cooking brownies. Typically, each sequence contains 500-1000 frames. For each instance we computed the quaternions for the 20 joints resulting in a 60 dimensional feature vector that describes the body configuration. CTW and LCTW are initialized as described in previous sections and optimized until convergence. The parameters of LCTW are manually set to $k_x = 3, k_y = 3$ and $p_w = .5, p_v = .3, p_r = .2$.

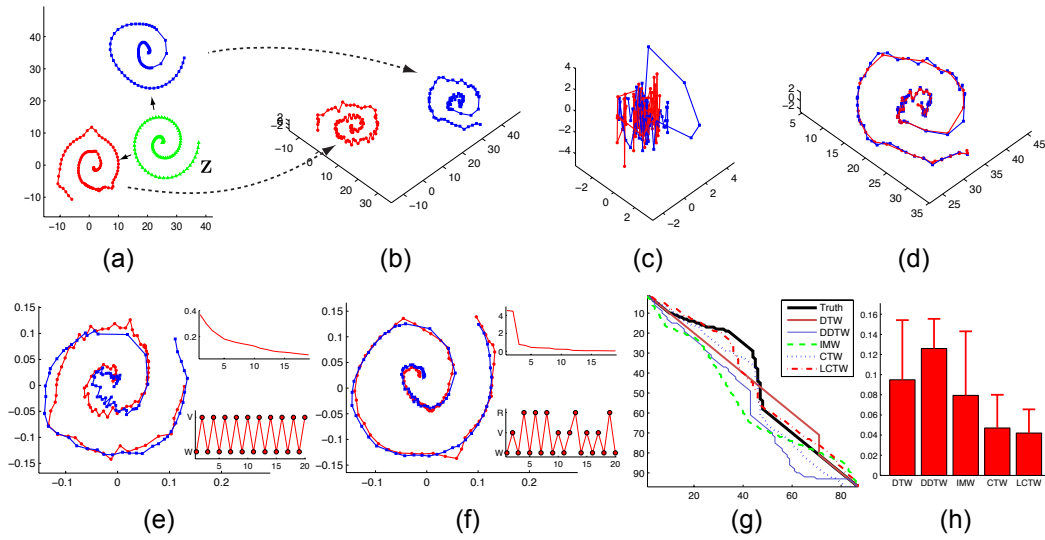

Figure 3: Example with synthetic data. Time series are generated by (a) spatio-temporal transformation of 2-D latent sequence (b) adding Gaussian noise in the $3^{rd}$ dimension. The result of space warping is computed by (c) derivative dynamic time warping (DDTW), (d) iterative time warping (IMW), (e) canonical time warping (CTW) and (f) local canonical time warping (LCTW). The energy function and order of optimizing the parameters for CTW and LCTW are shown in the top right and lower right corner of the graphs. (g) Comparison of the alignment results for several methods. (h) Mean and variance of the alignment error.

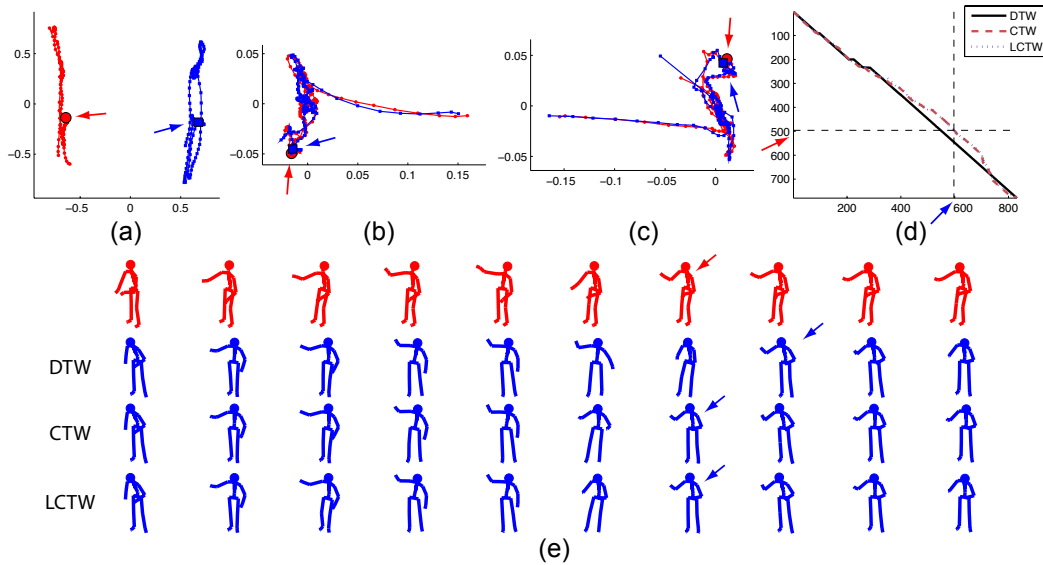

Figure 4: Example of motion capture data alignment. (a) PCA. (b) CTW. (c) LCTW. (d) Alignment path. (e) Motion capture data. $1^{st}$ row subject one, rest of the rows aligned subject two.

Fig. 4 shows the alignment results for the action of opening a cabinet. The projection on the principal components for both sequences can be seen in Fig. 4a. CTW and LCTW project the sequences in a low dimensional space that maximizes the correlation (Fig. 4b-c). Fig. 4d shows the alignment path. In this case, we do not have ground truth data, and we evaluated the results visually. The first row of Fig. 4e shows few instances of the first subject, and the last three rows the alignment of the third subject for DTW, CTW and LCTW. Observe that CTW and LCTW achieve better temporal alignment.

### 5.3 Facial expression data

In this experiment we tested the ability of CTW and LCTW to align facial expressions. We took 29 subjects from the RU-FACS database [27] which consists of interviews with men and women of varying ethnicity. The action units (AUs) in this database have been manually coded, and we selected AU12 (smiling) to run our experiments. Each event of AU12 is coded with an onset (start), peak and offset (end). We used person-specific AAM [28] to track 66 landmark points on the face. For the alignment of AU12 we only used 18 landmarks corresponding to the outline of the mouth, so for each frame we have a vector ($\mathbb{R}^{36 \times 1}$) with $(x, y)$ coordinates.

We took subject 14 and 30 and ran CTW and LCTW on the segments where the AU12 was coded. The parameters of LCTW are manually set to $k_x = 3, k_y = 3$ and $p_w = .5, p_v = .3, p_r = .2$. Fig. 5 shows the results of the alignment. Fig. 5b-c shows that the low dimensional projection obtained with CTW and LCTW has better alignment than DTW in Fig. 5a. Fig. 5d shows the position of the peak frame as the intersection of the two dotted lines. As we can observe from Fig. 5d, the alignment paths found by CTW and LCTW are closer to the manually labeled peak than the ones found by DTW. This shows that CTW and LCTW provide better alignment because the manually labeled peaks in both sequences should be aligned. Fig. 5e shows several frames illustrating the alignment.

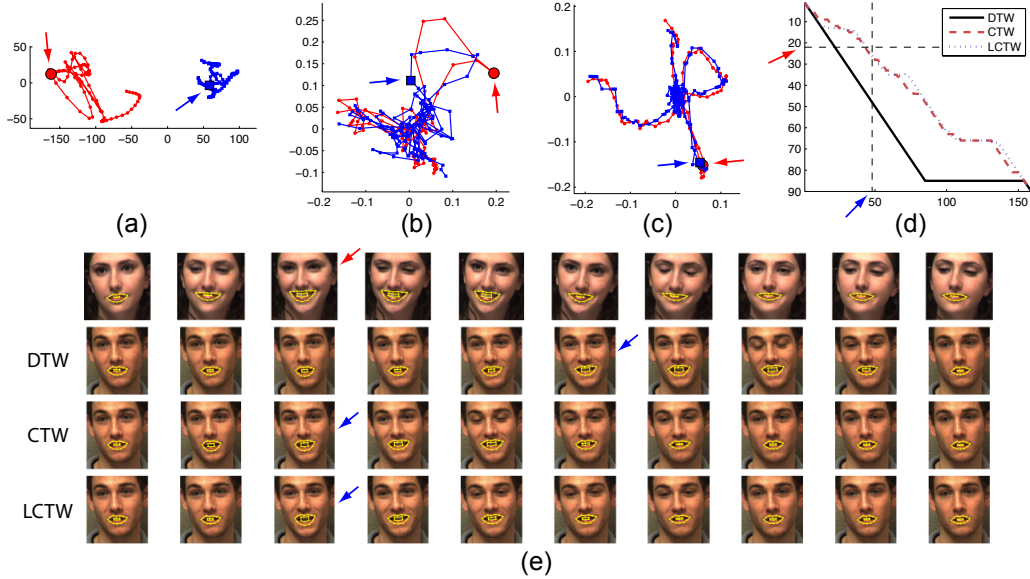

Figure 5: Example of facial expression alignment. (a) PCA. (b) CTW. (c) LCTW. (d) Alignment path. (e) Frames from an AU12 event. The AU peaks are indicated by arrows.

## 6   Conclusions

In this paper we proposed CTW and LCTW for spatio-temporal alignment of time series. CTW integrates the benefits of DTW and CCA into a clean and simple formulation. CTW extends DTW by adding a feature selection mechanism and enables alignment of signals with different dimensionality. CTW extends CCA by adding temporal alignment and allowing temporal local projections. We illustrated the benefits of CTW for alignment of motion capture data and facial expressions.

## 7   Acknowledgements

This material is based upon work partially supported by the National Science Foundation under Grant No. EEC-0540865.

## Footnotes

[1]Bold capital letters denote a matrix $\mathbf{X}$, bold lower-case letters a column vector $\mathbf{x}$. $\mathbf{x}_i$ represents the $i^{th}$ column of the matrix $\mathbf{X}$. $x_{ij}$ denotes the scalar in the $i^{th}$ row and $j^{th}$ column of the matrix $\mathbf{X}$. All non-bold letters represent scalars. $\mathbf{1}_{m \times n}, \mathbf{0}_{m \times n} \in \mathbb{R}^{m \times n}$ are matrices of ones and zeros. $\mathbf{I}_n \in \mathbb{R}^{n \times n}$ is an identity matrix. $\|\mathbf{x}\| = \sqrt{\mathbf{x}^T \mathbf{x}}$ denotes the Euclidean distance. $\|\mathbf{X}\|_F^2 = \text{Tr}(\mathbf{X}^T \mathbf{X})$ designates the Frobenious norm. $\mathbf{X} \circ \mathbf{Y}$ and $\mathbf{X} \otimes \mathbf{Y}$ are the Hadamard and Kronecker product of matrices. $\text{Vec}(\mathbf{X})$ denotes the vectorization of matrix $\mathbf{X}$. $\{i : j\}$ lists the integers, $\{i, i+1, \cdots, j-1, j\}$.

[2]The generation of time transformation matrix $\mathbf{M}_x$ (similar for $\mathbf{M}_y$) is initialized by setting $\mathbf{M}_x = \mathbf{I}_{n_x}$. Then, randomly pick and replicate $m - n_x$ columns of $\mathbf{M}_x$. We normalize each row $\mathbf{M}_x \mathbf{1}_m = \mathbf{1}_{n_x}$ to make the new frame to be an interpolation of $\mathbf{z}_i$.

# References

[1] I. N. Junejo, E. Dexter, I. Laptev, and P. Pérez. Cross-view action recognition from temporal self-similarities. In *ECCV*, pages 293–306, 2008.

[2] F. Zhou, F. de la Torre, and J. K. Hodgins. Aligned cluster analysis for temporal segmentation of human motion. In *FGR*, pages 1–7, 2008.

[3] E. Hsu, K. Pulli, and J. Popovic. Style translation for human motion. In *SIGGRAPH*, 2005.

[4] M. Brand, N. Oliver, and A. Pentland. Coupled hidden Markov models for complex action recognition. In *CVPR*, pages 994–999, 1997.

[5] M. Brand and A. Hertzmann. Style machines. In *SIGGRAPH*, pages 183–192, 2000.

[6] G. W. Taylor, G. E. Hinton, and S. T. Roweis. Modeling human motion using binary latent variables. In *NIPS*, volume 19, page 1345, 2007.

[7] A. Heloir, N. Courty, S. Gibet, and F. Multon. Temporal alignment of communicative gesture sequences. *J. Visual. Comp. Animat.*, 17(3-4):347–357, 2006.

[8] A. Shapiro, Y. Cao, and P. Faloutsos. Style components. In *Graphics Interface*, pages 33–39, 2006.

[9] G. Liu, Z. Pan, and Z. Lin. Style subspaces for character animation. *J. Visual. Comp. Animat.*, 19(3-4):199–209, 2008.

[10] A. M. Elgammal and C.-S. Lee. Separating style and content on a nonlinear manifold. In *CVPR*, 2004.

[11] Y. Caspi and M. Irani. Aligning non-overlapping sequences. *Int. J. Comput. Vis.*, 48(1):39–51, 2002.

[12] C. Rao, A. Gritai, M. Shah, and T. Fathima Syeda-Mahmood. View-invariant alignment and matching of video sequences. In *ICCV*, pages 939–945, 2003.

[13] A. Gritai, Y. Sheikh, C. Rao, and M. Shah. Matching trajectories of anatomical landmarks under viewpoint, anthropometric and temporal transforms. *Int. J. Comput. Vis.*, 2009.

[14] L. Rabiner and B.-H. Juang. *Fundamentals of speech recognition*. Prentice Hall, 1993.

[15] E. J. Keogh and M. J. Pazzani. Derivative dynamic time warping. In *SIAM ICDM*, 2001.

[16] J. Listgarten, R. M. Neal, S. T. Roweis, and A. Emili. Multiple alignment of continuous time series. In *NIPS*, pages 817–824, 2005.

[17] Y. Sheikh, M. Sheikh, and M. Shah. Exploring the space of a human action. In *ICCV*, 2005.

[18] T. W. Anderson. *An introduction to multivariate statistical analysis*. Wiley-Interscience, 2003.

[19] F. de la Torre. A unification of component analysis methods. *Handbook of Pattern Recognition and Computer Vision*, 2009.

[20] T. K. Kim and R. Cipolla. Canonical correlation analysis of video volume tensors for action categorization and detection. *IEEE Trans. Pattern Anal. Mach. Intell.*, 31:1415–1428, 2009.

[21] C. C. Loy, T. Xiang, and S. Gong. Multi-camera activity correlation analysis. In *CVPR*, 2009.

[22] B. Fischer, V. Roth, and J. Buhmann. Time-series alignment by non-negative multiple generalized canonical correlation analysis. *BMC bioinformatics*, 8(10), 2007.

[23] D. P. Bertsekas. *Dynamic programming and optimal control*. 1995.

[24] Z. Ghahramani and G. E. Hinton. The EM algorithm for mixtures of factor analyzers. *University of Toronto Tec. Rep.*, 1997.

[25] J. J. Verbeek, S. T. Roweis, and N. A. Vlassis. Non-linear CCA and PCA by alignment of local models. In *NIPS*, 2003.

[26] F. de la Torre, J. K. Hodgins, J. Montano, S. Valcarcel, A. Bargteil, X. Martin, J. Macey, A. Collado, and P. Beltran. Guide to the Carnegie Mellon University Multimodal Activity (CMU-MMAC) Database. *Carnegie Mellon University Tec. Rep.*, 2009.

[27] M. S. Bartlett, G. C. Littlewort, M. G. Frank, C. Lainscsek, I. Fasel, and J. R. Movellan. Automatic recognition of facial actions in spontaneous expressions. *J. Multimed.*, 1(6):22–35, 2006.

[28] I. Matthews and S. Baker. Active appearance models revisited. *Int. J. Comput. Vis.*, 60(2):135–164, 2004.

